# Efficient Exact Inference in Planar Ising Models

**Nicol N. Schraudolph**
nips@schraudolph.org

**Dmitry Kamenetsky**
dkamen@cecs.anu.edu.au

National ICT Australia, Locked Bag 8001, Canberra ACT 2601, Australia
& RSISE, Australian National University, Canberra ACT 0200, Australia

## Abstract

We give polynomial-time algorithms for the exact computation of lowest-energy states, worst margin violators, partition functions, and marginals in certain binary undirected graphical models. Our approach provides an interesting alternative to the well-known graph cut paradigm in that it does not impose any submodularity constraints; instead we require planarity to establish a correspondence with perfect matchings in an expanded dual graph. Maximum-margin parameter estimation for a boundary detection task shows our approach to be efficient and effective. A C++ implementation is available from http://nic.schraudolph.org/isinf/.

## 1  Introduction

Undirected graphical models are a popular tool in machine learning; they represent real-valued energy functions of the form

$$E'(\boldsymbol{y}) \ := \ \sum_{i \in \mathcal{V}} E'_i(y_i) \ + \ \sum_{(i,j) \in \mathcal{E}} E'_{ij}(y_i, y_j)\,, \tag{1}$$

where the terms in the first sum range over the nodes $\mathcal{V} = \{1, 2, \ldots n\}$, and those in the second sum over the edges $\mathcal{E} \subseteq \mathcal{V} \times \mathcal{V}$ of an undirected graph $G(\mathcal{V}, \mathcal{E})$.

The *junction tree* decomposition provides an efficient framework for exact statistical inference in graphs that are (or can be turned into) trees of small cliques. The resulting algorithms, however, are exponential in the clique size, *i.e.*, the *treewidth* of the original graph. This is prohibitively large for many graphs of practical interest — for instance, it grows as $O(n)$ for an $n \times n$ square lattice. Many approximate inference techniques have been developed so as to deal with such graphs, such as pseudo-likelihood, mean field approximation, loopy belief propagation, and tree reweighting.

### 1.1  The Ising Model

Efficient *exact* inference is possible in certain graphical models with binary node labels. Here we focus on *Ising* models, whose energy functions have the form $E : \{0, 1\}^n \to \mathbb{R}$ with

$$E(\boldsymbol{y}) \ := \ \sum_{(i,j) \in \mathcal{E}} [y_i \neq y_j]\, E_{ij}, \tag{2}$$

where $[\cdot]$ denotes the indicator function, *i.e.*, the cost $E_{ij}$ is incurred only in those states $\boldsymbol{y}$ where $y_i$ and $y_j$ disagree. Compared to the general model (1) for binary nodes, (2) imposes two additional restrictions: zero node energies, and edge energies in the form of disagreement costs. At first glance these constraints look severe; for instance, such systems must obey the symmetry $E(\boldsymbol{y}) = E(\neg \boldsymbol{y})$, where $\neg$ denotes Boolean negation (ones' complement). It is well known, however, that adding a single node makes the Ising model (2) as expressive as the general model (1) for binary variables:

**Theorem 1** *Every energy function of the form* (1) *over $n$ binary variables is equivalent to an Ising energy function of the form* (2) *over $n + 1$ variables, with the additional variable held constant.*

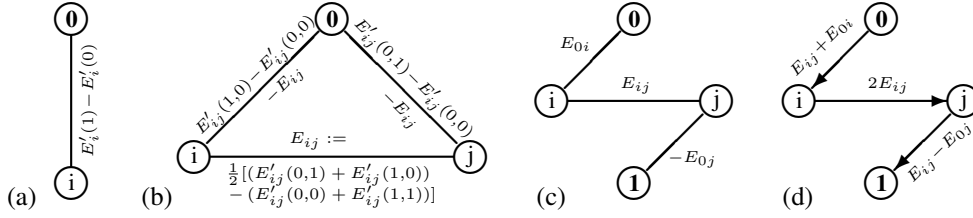

Figure 1: Equivalent Ising model (with disagreement costs) for a given (a) node energy $E'_i$, (b) edge energy $E'_{ij}$ in a binary graphical model; (c) equivalent submodular model if $E_{ij} > 0$ and $E_{0i} > 0$ but $E_{0j} < 0$; (d) equivalent directed model of Kolmogorov and Zabih [1], Fig. 2d.

**Proof** by construction: Two energy functions are equivalent if they differ only by a constant. Without loss of generality, denote the additional variable $y_0$ and hold it constant at $y_0 := 0$. Given an energy function of the form (1), construct an Ising model with disagreement costs as follows:

1. For each node energy function $E'_i(y_i)$, add a disagreement cost term $E_{0i} := E'_i(1) - E'_i(0)$;

2. For each edge energy function $E'_{ij}(y_i, y_j)$, add the three disagreement cost terms

$$E_{ij} := \tfrac{1}{2}[(E'_{ij}(0,1) + E'_{ij}(1,0)) - (E'_{ij}(0,0) + E'_{ij}(1,1))],$$
$$E_{0i} := E'_{ij}(1,0) - E'_{ij}(0,0) - E_{ij}, \quad \text{and} \tag{3}$$
$$E_{0j} := E'_{ij}(0,1) - E'_{ij}(0,0) - E_{ij}.$$

Summing the above terms, the total *bias* of node $i$ (*i.e.*, its disagreement cost with the bias node) is

$$E_{0i} = E'_i(1) - E'_i(0) + \sum_{j:(i,j)\in\mathcal{E}} [E'_{ij}(1,0) - E'_{ij}(0,0) - E_{ij}]. \tag{4}$$

This construction defines an Ising model whose energy in every configuration $\boldsymbol{y}$ is shifted, relative to that of the general model we started with, by the same constant amount, namely $E'(\boldsymbol{0})$:

$$\forall \boldsymbol{y} \in \{0,1\}^n : \quad E\left(\begin{bmatrix} 0 \\ \boldsymbol{y} \end{bmatrix}\right) = E'(\boldsymbol{y}) - \sum_{i\in\mathcal{V}} E'_i(0) - \sum_{(i,j)\in\mathcal{E}} E'_{ij}(0,0) = E'(\boldsymbol{y}) - E'(\boldsymbol{0}). \tag{5}$$

The two models' energy functions are therefore equivalent. ∎

Note how in the above construction the label symmetry $E(\boldsymbol{y}) = E(\neg \boldsymbol{y})$ of the plain Ising model (2) is conveniently broken by the introduction of a bias node, through the convention that $y_0 := 0$.

## 1.2 Energy Minimization via Graph Cuts

**Definition 2** *The* cut $\mathcal{C}$ *of a binary graphical model* $G(\mathcal{V}, \mathcal{E})$ *induced by state* $\boldsymbol{y} \in \{0,1\}^n$ *is the set* $\mathcal{C}(\boldsymbol{y}) := \{(i,j) \in \mathcal{E} : y_i \neq y_j\}$; *its* weight $|\mathcal{C}(\boldsymbol{y})|$ *is the sum of the weights of its edges.*

Any given state $\boldsymbol{y}$ partitions the nodes of a binary graphical model into two sets: those labeled '0', and those labeled '1'. The corresponding *graph cut* is the set of edges crossing the partition; since only they contribute disagreement costs to the Ising model (2), we have $\forall \boldsymbol{y} : |\mathcal{C}(\boldsymbol{y})| = E(\boldsymbol{y})$. The lowest-energy state of an Ising model therefore induces its minimum-weight cut.

Minimum-weight cuts can be computed in polynomial time in graphs whose edge weights are all non-negative. Introducing one more node, with the constraint $y_{n+1} := 1$, allows us to construct an equivalent energy function by replacing each negatively weighted bias edge $E_{0i} < 0$ by an edge to the new node $n+1$ with the positive weight $E_{i,n+1} := -E_{0i} > 0$ (Figure 1c). This still leaves us with the requirement that all non-bias edges be non-negative. This *submodularity* constraint implies that agreement between nodes must be locally preferable to disagreement — a severe limitation.

Graph cuts have been widely used in machine learning to find lowest-energy configurations, in particular in image processing. Our construction (Figure 1c) differs from that of Kolmogorov and Zabih [1] (Figure 1d) in that we do not employ the notion of *directed* edges. (In directed graphs, the weight of a cut is the sum of the weights of only those edges crossing the cut in a given direction.)

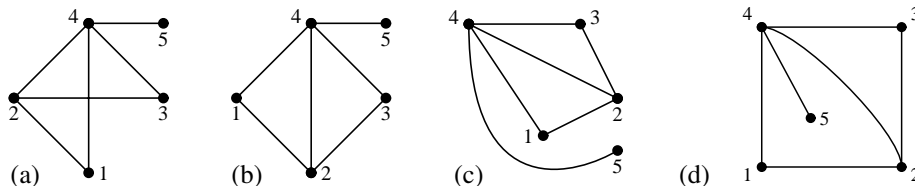

Figure 2: (a) a non-plane drawing of a planar graph; (b) a plane drawing of the same graph; (c) a different plane drawing of same graph, with the same planar embedding as (b); (d) a plane drawing of the same graph with a different planar embedding.

## 2   Planarity

Unlike graph cut methods, the inference algorithms we describe below do not depend on submodularity; instead they require that the model graph be *planar*, and that a planar *embedding* be provided.

### 2.1   Embedding Planar Graphs

**Definition 3** *Let $G(\mathcal{V}, \mathcal{E})$ be an undirected, connected graph. For each vertex $i \in \mathcal{V}$, let $\mathcal{E}_i$ denote the set of edges in $\mathcal{E}$ incident upon $i$,* considered as being oriented away from $i$, *and let $\pi_i$ be a cyclic permutation of $\mathcal{E}_i$. A rotation system* for G *is a set of permutations $\Pi = \{\pi_i : i \in \mathcal{V}\}$.*

Rotation systems [2] directly correspond to topological graph embeddings in orientable surfaces:

**Theorem 4** (White and Beineke [2], p. 22f) *Each rotation system determines an embedding of G in some orientable surface S such that $\forall i \in \mathcal{V}$, any edge $(i, j) \in \mathcal{E}_i$ is followed by $\pi_i(i, j)$ in (say) clockwise orientation, and such that the* faces $\mathcal{F}$ *of the embedding, given by the orbits of the mapping $(i, j) \to \pi_j(j, i)$, are 2-cells (topological disks).*

Note that while in graph visualisation "embedding" is often used as a synonym for "drawing", in modern topological graph theory it stands for "rotation system". We adopt the latter usage, which views embeddings as equivalence classes of graph drawings characterized by identical cyclic ordering of the edges incident upon each vertex. For instance, $\pi_4(4, 5) = (4, 3)$ in Figures 2b and 2c (same embedding) but $\pi_4(4, 5) = (4, 1)$ in Figure 2d (different embedding). A sample face in Figures 2b–2d is given by the orbit $(4, 1) \to \pi_1(1, 4) = (1, 2) \to \pi_2(2, 1) = (2, 4) \to \pi_4(4, 2) = (4, 1)$. The *genus g* of the embedding surface $S$ can be determined from the *Euler characteristic*

$$|\mathcal{V}| - |\mathcal{E}| + |\mathcal{F}| = 2 - 2g, \tag{6}$$

where $|\mathcal{F}|$ is found by counting the orbits of the rotation system, as described in Theorem 4. Since planar graphs are exactly those that can be embedded on a surface of genus $g = 0$ (a topological sphere), we arrive at a purely combinatorial definition of planarity:

**Definition 5** *A graph $G(\mathcal{V}, \mathcal{E})$ is* planar *iff it has a rotation system $\Pi$ producing exactly $2 + |\mathcal{E}| - |\mathcal{V}|$ orbits. Such a system is called a* planar embedding *of G, and $G(\mathcal{V}, \mathcal{E}, \Pi)$ is called a* plane graph.

Our inference algorithms require a plane graph as input. In certain domains (*e.g.*, when working with geographic information) a plane drawing of the graph (from which the corresponding embedding is readily determined) may be available. Where it is not, we employ the algorithm of Boyer and Myrvold [3] which, given any connected graph $G$ as input, produces in linear time either a planar embedding for $G$ or a proof that $G$ is non-planar. Source code for this step is freely available [3, 4].

### 2.2   The Planarity Constraint

In Section 1.1 we have mapped a general binary graphical model to an Ising model with an additional bias node; now we require that that Ising model be planar. What does that imply for the original, general model? If all nodes of the graph are to be connected to the bias node without violating planarity, the graph has to be *outerplanar*, *i.e.*, have a planar embedding in which all its nodes lie on the external face — a very severe restriction.

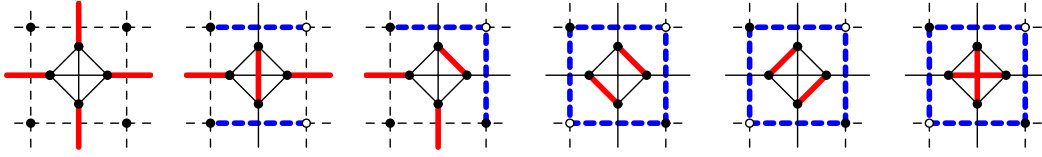

Figure 3: Possible cuts (bold blue dashes) of a square face of the model graph (dashed) and complementary perfect matchings (bold red lines) of its expanded dual (solid lines).

The situation improves, however, if only a subset $\mathcal{B} \subset \mathcal{V}$ of nodes have non-zero bias (4): Then the graph only has to be $\mathcal{B}$-outerplanar, *i.e.*, have a planar embedding in which all nodes in $\mathcal{B}$ lie on the same face. In image processing, for instance, where it is common to operate on a square grid of pixels, we can permit bias for all nodes on the perimeter of the grid. In general, a planar embedding which maximizes a weighted sum over the nodes bordering a given face can be found in linear time [5]; by setting node weights to some measure of bias (such as $E_{0_i}{}^2$) we can efficiently obtain the planar Ising model closest (in that measure) to any given planar binary graphical model.

In contrast to submodularity, $\mathcal{B}$-outerplanarity is a structural constraint. This has the advantage that once a model obeying the constraint is selected, inference (*e.g.*, parameter estimation) can proceed via unconstrained methods (*e.g.*, optimization). Finally, we note that all our algorithms can be extended to work for non-planar graphs as well. They then take time exponential in the genus of the embedding though still polynomial in the size of the graph; for graphs of low genus this may well be preferable to current approximative methods.

## 3  Computing Optimal States via Maximum-Weight Perfect Matching

The relationship between the states of a planar Ising model and perfect matchings ("dimer coverings" to physicists) was first discovered by Kasteleyn [6] and Fisher [7]. Globerson and Jaakkola [8] presented a more direct construction for triangulated graphs, which we generalize here.

### 3.1  The Expanded Dual Graph

**Definition 6** *The* dual $G^*(\mathcal{F}, \mathcal{E})$ *of an embedded graph* $G(\mathcal{V}, \mathcal{E}, \Pi)$ *has a vertex for each face of* $G$, *with edges connecting vertices corresponding to faces that are adjacent (*i.e., *share an edge) in* $G$.

Each edge of the dual crosses exactly one edge of the original graph; due to this one-to-one relationship we will consider the dual to have the *same* set of edges $\mathcal{E}$ (with the same energies) as the original.

We now expand the dual graph by replacing each node with a $q$-clique, where $q$ is the degree of the node, as shown in Figure 3 for $q = 4$. The additional edges internal to each $q$-clique are given zero energy so as to leave the model unaffected. For large $q$ the introduction of these $O(q^2)$ internal edges slows down subsequent computations (solid line in Figure 4, left); this can be avoided by subdividing the offending $q$-gonal face with chords (which are also given zero energy) before constructing the dual. Our implementation performs best when "octangulating" the graph, *i.e.*, splitting octagons off all faces with $q > 13$; this is more efficient than a full triangulation (Figure 4, left).

### 3.2  Complementary Perfect Matchings

**Definition 7** *A* perfect matching *of a graph* $G(\mathcal{V}, \mathcal{E})$ *is a subset* $\mathcal{M} \subseteq \mathcal{E}$ *of edges wherein exactly one edge is incident upon each vertex in* $\mathcal{V}$; *its* weight $|\mathcal{M}|$ *is the sum of the weights of its edges.*

**Theorem 8** *For every cut* $\mathcal{C}$ *of an embedded graph* $G(\mathcal{V}, \mathcal{E}, \Pi)$ *there exists at least one (if* $G$ *is triangulated: exactly one) perfect matching* $\mathcal{M}$ *of its expanded dual* complementary *to* $\mathcal{C}$, i.e., $\mathcal{E} \backslash \mathcal{M} = \mathcal{C}$.

**Proof sketch** Consider the complement $\mathcal{E} \backslash \mathcal{C}$ of the cut as a partial matching of the expanded dual. By definition, $\mathcal{C}$ intersects any cycle of $G$, and therefore also the perimeters of $G$'s faces $\mathcal{F}$, in an even number of edges. In each clique of the expanded dual, $\mathcal{C}$'s complement thus leaves an even

number of nodes unmatched; $\mathcal{M}$ can therefore be completed using only edges interior to the cliques. In a 3-clique, there is only one way to do this, so $\mathcal{M}$ is unique if $G$ is triangulated. ∎

In other words, there exists a surjection from perfect matchings in the expanded dual of $G$ to cuts in $G$. Furthermore, since we have given edges interior to the cliques of the expanded dual zero energy, every perfect matching $\mathcal{M}$ complementary to a cut $\mathcal{C}$ of our Ising model (2) obeys the relation

$$|\mathcal{M}| + |\mathcal{C}| = \sum_{(i,j) \in \mathcal{E}} E_{ij} = \text{const.} \tag{7}$$

This means that instead of a minimum-weight cut in a graph we can look for a maximum-weight perfect matching in its expanded dual. But will that matching always be complementary to a cut?

**Theorem 9** *Every perfect matching $\mathcal{M}$ of the expanded dual of a* plane *graph $G(\mathcal{V}, \mathcal{E}, \Pi)$ is complementary to a cut $\mathcal{C}$ of $G$, i.e., $\mathcal{E} \backslash \mathcal{M} = \mathcal{C}$.*

**Proof sketch** In each clique of the expanded dual, an even number of nodes is matched by edges interior to the clique. The complement $\mathcal{E} \backslash \mathcal{M}$ of the matching in $G$ thus contains an even number of edges around the perimeter of each face of the embedding. By induction over faces, this holds for every *contractible* (on the embedding surface) cycle of $G$. Because a plane is simply connected, all cycles in a plane graph are contractible; thus $\mathcal{E} \backslash \mathcal{M}$ is a cut. ∎

This is where planarity matters: Surfaces of non-zero genus are not simply connected, and thus non-plane graphs may contain non-contractible cycles; our construction does not guarantee that the complement $\mathcal{E} \backslash \mathcal{M}$ of a perfect matching of the expanded dual contains an even number of edges along such cycles. For planar graphs, however, the above theorems allow us to leverage known polynomial-time algorithms for perfect matchings into inference methods for Ising models.

## 3.3 The Lowest-Energy (MAP or Ground) State

The blossom-shrinking algorithm [9, 10] is a sophisticated method to efficiently compute the maximum-weight perfect matching of a graph. It can be implemented to run in as little as $O(|\mathcal{E}||\mathcal{V}|\log|\mathcal{V}|)$ time. Although the Blossom IV code we are using [11] is asymptotically less efficient — $O(|\mathcal{E}||\mathcal{V}|^2)$ — we have found it to be very fast in practice (Figure 4, left).

We can now efficiently compute the lowest-energy state of a planar Ising model as follows: Find a planar embedding of the model graph (Section 2.1), construct its expanded dual (Section 3.1), and run the blossom-shrinking algorithm on that to compute its maximum-weight perfect matching. Its complement in the original model is the minimum-weight graph cut (Section 3.2). We can identify the state which induces this cut via a depth-first graph traversal that labels nodes as it encounters them, starting by labeling the bias node $y_0 := 0$; this is shown below as Algorithm 1.

---
**Algorithm 1** Find State from Corresponding Graph Cut

| | | |
|---|---|---|
| **Input:** | Ising model graph $G(\mathcal{V}, \mathcal{E})$ | **procedure** dfs_state($i \in \{0, 1, 2, \ldots n\}, s \in \{0, 1\}$) |
| | graph cut $\mathcal{C}(\boldsymbol{y}) \subseteq \mathcal{E}$ | **if** $y_i = $ unknown **then** |
| 1. | $\forall i \in \{0, 1, 2, \ldots n\}$ : | $\quad y_i := s; \ \forall(i,j) \in \mathcal{E}_i$ : |
| | $\quad y_i := $ unknown; | $\quad\quad$ **if** $(i,j) \in \mathcal{C}$ **then** dfs_state($j, \neg s$); |
| 2. | dfs_state(0, 0); | $\quad\quad$ **else** dfs_state($j, s$); |
| **Output:** | state vector $\boldsymbol{y}$ | **else assert** $y_i = s$; |

---

## 3.4 The Worst Margin Violator

Maximum-margin parameter estimation in graphical models involves determining the *worst margin violator* — the state that minimizes, relative to a given target state $\boldsymbol{y}^*$, the *margin energy*

$$M(\boldsymbol{y}|\boldsymbol{y}^*) := E(\boldsymbol{y}) - d(\boldsymbol{y}|\boldsymbol{y}^*), \tag{8}$$

where $d(\cdot|\cdot)$ is a measure of divergence in state space. If $d(\cdot|\cdot)$ is the weighted Hamming distance

$$d(\boldsymbol{y}|\boldsymbol{y}^*) := \sum_{(i,j) \in \mathcal{E}} [[y_i \neq y_j] \neq [y_i^* \neq y_j^*]] v_{ij}, \tag{9}$$

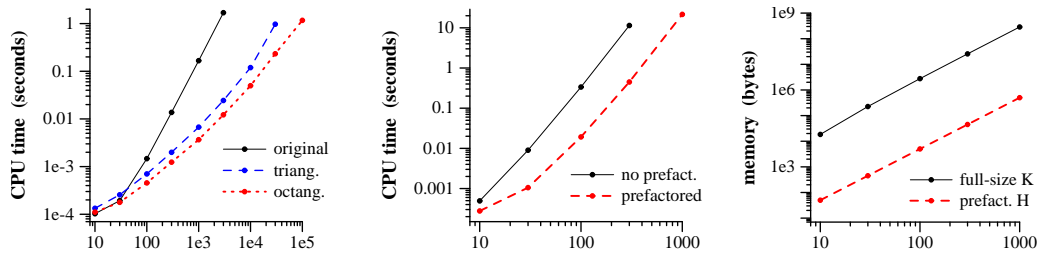

Figure 4: Cost of inference on a ring graph, plotted against ring size. Left & center: CPU time on Apple MacBook with 2.2 GHz Intel Core2 Duo processor; right: storage size. Left: MAP state via Blossom IV [11] on original, triangulated, and octangulated ring; center & right: marginal probabilities, full matrix $\boldsymbol{K}$ (double precision, no prefactoring) *vs.* prefactored half-Kasteleyn bitmatrix $\boldsymbol{H}$.

where the $v_{ij} \geq 0$ are constant weighting factors (in the simplest case: all ones) on the edges of our Ising model, then it is easily verified that the margin energy (8) is implemented (up to a shift that depends only on $\boldsymbol{y}^*$) by an isomorphic Ising model with disagreement costs

$$ E_{ij} + (2\,[y_i^* \neq y_j^*] - 1)\,v_{ij}. \tag{10} $$

We can thus use our algorithm of Section 3.3 to efficiently find the worst margin violator, $\operatorname{argmin}_{\boldsymbol{y}} M(\boldsymbol{y}|\boldsymbol{y}^*)$, for maximum-margin parameter estimation.

## 4   Computing the Partition Function and Marginal Probabilities[1]

A Markov random field (MRF) over our Ising model (2) models the distribution

$$ \mathbb{P}(\boldsymbol{y}) = \tfrac{1}{Z}\,e^{-E(\boldsymbol{y})}, \quad \text{where} \quad Z := \sum_{\boldsymbol{y}} e^{-E(\boldsymbol{y})} \tag{11} $$

is the MRF's *partition function*. As it involves a summation over exponentially many states $\boldsymbol{y}$, calculating the partition function is generally intractable. For planar graphs, however, the generating function for perfect matchings can be calculated in polynomial time via the determinant of a skew-symmetric matrix [6, 7]. Due to the close relationship with graph cuts (Section 3.2) we can calculate $Z$ in (11) likewise. Elaborating on work of Globerson and Jaakkola [8], we first convert the Ising model graph into a Boolean "half-Kasteleyn" matrix $\boldsymbol{H}$:

1. *plane triangulate* the embedded graph so as to make the relationship between cuts and complementary perfect matchings a bijection (*cf.* Section 3.2);
2. *orient* the edges of the graph such that the in-degree of every node is odd;
3. *construct* the Boolean half-Kasteleyn matrix $\boldsymbol{H}$ from the oriented graph;
4. *prefactor* the triangulation edges (added in Step 1) out of $\boldsymbol{H}$.

Our Step 2 simplifies equivalent operations in previous constructions [6–8], Step 3 differs in that it only sets unit (*i.e.*, +1) entries in a Boolean matrix, and Step 4 can dramatically reduce the size of $\boldsymbol{H}$ for compact storage (as a bit matrix) and faster subsequent computations (Figure 4, center & right).

For a given set of disagreement edge costs $E_k, k = \{1, 2, , \ldots |\mathcal{E}|\}$ on that graph, we then build from $\boldsymbol{H}$ and the $E_k$ the skew-symmetric, real-valued Kasteleyn matrix $\boldsymbol{K}$:

1. $\boldsymbol{K} := \boldsymbol{H}$;
2. $\forall k \in \{1, 2, , \ldots |\mathcal{E}|\} : \boldsymbol{K}_{2k-1,2k} := \boldsymbol{K}_{2k-1,2k} + e^{E_k}$;
3. $\boldsymbol{K} := \boldsymbol{K} - \boldsymbol{K}^\top$.

The partition function for perfect matchings is $\sqrt{|\boldsymbol{K}|}$ [6–8], so we factor $\boldsymbol{K}$ and use (7) to compute the log partition function for (11) as $\ln Z = \tfrac{1}{2}\ln|\boldsymbol{K}| - \sum_{k\in\mathcal{E}} E_k$. Its derivative yields the *marginal probability* of disagreement on the $k^{\text{th}}$ edge, and is computed via the inverse of $\boldsymbol{K}$:

$$ \mathbb{P}(k \in \mathcal{C}) := -\frac{\partial \ln Z}{\partial E_k} = 1 - \frac{1}{2|\boldsymbol{K}|}\frac{\partial|\boldsymbol{K}|}{\partial E_k} = 1 - \tfrac{1}{2}\operatorname{tr}\left(\boldsymbol{K}^{-1}\frac{\partial \boldsymbol{K}}{\partial E_k}\right) = 1 + \boldsymbol{K}^{-1}_{2k-1,2k}\,\boldsymbol{K}_{2k-1,2k}. $$

$$ \tag{12} $$

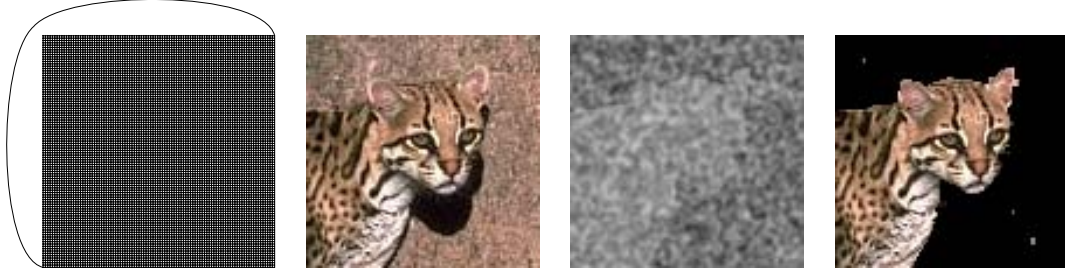

Figure 5: Boundary detection by maximum-margin training of planar Ising grids; from left to right: Ising model ($100 \times 100$ grid), original image, noisy mask, and MAP segmentation of the Ising grid.

## 5  Maximum Likelihood *vs.* Maximum Margin CRF Parameter Estimation

Our algorithms can be applied to regularized parameter estimation in conditional random fields (CRFs). In a linear planar Ising CRF, the disagreement costs $E_{ij}$ in (2) are computed as inner products between features (sufficient statistics) $\boldsymbol{x}$ of the modeled data and corresponding parameters $\boldsymbol{\theta}$ of the model, and (11) is used to model the *conditional* distribution $\mathbb{P}(\boldsymbol{y}|\boldsymbol{x}, \boldsymbol{\theta})$. Maximum likelihood (ML) parameter estimation then seeks to minimize *wrt.* $\boldsymbol{\theta}$ the $L_2$-regularized negative log likelihood

$$L_{\mathrm{ML}}(\boldsymbol{\theta}) := \tfrac{1}{2}\lambda\|\boldsymbol{\theta}\|^2 - \ln\mathbb{P}(\boldsymbol{y}^*|\boldsymbol{x}, \boldsymbol{\theta}) = \tfrac{1}{2}\lambda\|\boldsymbol{\theta}\|^2 + E(\boldsymbol{y}^*|\boldsymbol{x}, \boldsymbol{\theta}) + \ln Z(\boldsymbol{\theta}|\boldsymbol{x}) \qquad (13)$$

of a given target labeling $\boldsymbol{y}^*$,[2] with regularization parameter $\lambda$. This is a smooth, convex objective that can be optimized via batch or online implementations of gradient methods such as LBFGS [13]; the gradient of the log partition function in (13) is obtained by computing the marginals (12). For maximum margin (MM) parameter estimation [14] we instead minimize

$$L_{\mathrm{MM}}(\boldsymbol{\theta}) := \tfrac{1}{2}\lambda\|\boldsymbol{\theta}\|^2 + E(\boldsymbol{y}^*|\boldsymbol{x}, \boldsymbol{\theta}) - \min_{\boldsymbol{y}} M(\boldsymbol{y}|\boldsymbol{y}^*, \boldsymbol{x}, \boldsymbol{\theta}) \qquad (14)$$
$$= \tfrac{1}{2}\lambda\|\boldsymbol{\theta}\|^2 + E(\boldsymbol{y}^*|\boldsymbol{x}, \boldsymbol{\theta}) - E(\hat{\boldsymbol{y}}|\boldsymbol{x}, \boldsymbol{\theta}) + d(\hat{\boldsymbol{y}}|\boldsymbol{y}^*),$$

where $\hat{\boldsymbol{y}} := \operatorname{argmin}_{\boldsymbol{y}} M(\boldsymbol{y}|\boldsymbol{y}^*, \boldsymbol{x}, \boldsymbol{\theta})$ is the worst margin violator, *i.e.*, the state that minimizes the margin energy (8). $L_{\mathrm{MM}}(\boldsymbol{\theta})$ is convex but non-smooth; we can minimize it via bundle methods such as the BT bundle trust algorithm [15], making use of the convenient lower bound $\forall \boldsymbol{\theta}: L_{\mathrm{MM}}(\boldsymbol{\theta}) \geq 0$.

To demonstrate the scalability of planar Ising models, we designed a simple boundary detection task based on images from the GrabCut Ground Truth image segmentation database [16]. We took $100 \times 100$ pixel subregions of images that depicted a segmentation boundary, and corrupted the segmentation mask with pink noise, produced by convolving a white noise image (all pixels i.i.d. uniformly random) with a Gaussian density with one pixel standard deviation. We then employed a planar Ising model to recover the original boundary — namely, a $100 \times 100$ square grid with one additional edge pegged to a high energy, encoding prior knowledge that two opposing corners of the grid depict different regions (Figure 5, left). The energy of the other edges was $E_{ij} := \langle [1, |x_i - x_j|], \boldsymbol{\theta} \rangle$, where $x_i$ is the pixel intensity at node $i$. We did not employ a bias node for this task, and simply set $\lambda = 1$.

Note that this is a huge model: 10 000 nodes and 19 801 edges. Computing the partition function or marginals would require inverting a Kasteleyn matrix with over $1.5 \cdot 10^9$ entries; minimizing (13) is therefore computationally infeasible for us. Computing a ground state via the algorithm described in Section 3, by contrast, takes only 0.3 seconds on an Apple MacBook with 2.2 GHz Intel Core2 Duo processor. We can therefore efficiently minimize (14) to obtain the MM parameter vector $\boldsymbol{\theta}^*$, then compute the CRF's MAP (*i.e.*, ground) state for rapid prediction.

Figure 5 (right) shows how even for a signal-to-noise (S/N) ratio of 1:8, our approach is capable of recovering the original segmentation boundary quite well, with only 0.67% of nodes mislabeled here. For S/N ratios of 1:9 and lower the system was unable to locate the boundary; for S/N ratios of 1:7 and higher we obtained perfect reconstruction. Further experiments are reported in [12]. On smaller grids, ML parameter estimation and marginals for prediction become computationally feasible, if slower than the MM/MAP approach. This will allow direct comparison of ML *vs.* MM for parameter estimation, and MAP *vs.* marginals for prediction, to our knowledge for the first time on graphs intractable for the junction tree apppproach, such as the grids often used in image processing.

# 6 Discussion

We have proposed an alternative algorithmic framework for efficient exact inference in binary graphical models, which replaces the submodularity constraint of graph cut methods with a planarity constraint. Besides proving efficient and effective in first experiments, our approach opens up a number of interesting research directions to be explored:

Our algorithms can all be extended to nonplanar graphs, at a cost exponential in the genus of the embedding. We are currently developing these extensions, which may prove of great practical value for graphs that are "almost" planar; examples include road networks (where edge crossings arise from overpasses without on-ramps) and graphs describing the tertiary structure of proteins [17]. These algorithms also provide a foundation for the future development of efficient *approximate* inference methods for nonplanar Ising models.

Our method for calculating the ground state (Section 3) actually works for nonplanar graphs whose ground state does not contain frustrated non-contractible cycles. The QPBO graph cut method [18] finds ground states that do not contain any frustrated cycles, and otherwise yields a *partial* labeling. Can we likewise obtain a partial labeling of ground states with frustrated non-contractible cycles?

The existence of two distinct tractable frameworks for inference in binary graphical models implies a yet more powerful hybrid: Consider a graph each of whose biconnected components is either planar or submodular. As a whole, this graph may be neither planar nor submodular, yet efficient exact inference in it is clearly possible by applying the appropriate framework to each component. Can this hybrid approach be extended to cover less obvious situations?

## Footnotes

[1] We only have space for a high-level overview here; see [12] for full details.

[2] For notational clarity we suppress here the fact that we are usually modeling a *collection* of data items.

## References

[1] V. Kolmogorov and R. Zabih. What energy functions can be minimized via graph cuts? *IEEE Trans. Pattern Analysis and Machine Intelligence*, 26(2):147–159, 2004.

[2] A. T. White and L. W. Beineke. Topological graph theory. In L. W. Beineke and R. J. Wilson, editors, *Selected Topics in Graph Theory*, chapter 2, pages 15–49. Academic Press, 1978.

[3] J. M. Boyer and W. J. Myrvold. On the cutting edge: Simplified $O(n)$ planarity by edge addition. *Journal of Graph Algorithms and Applications*, 8(3):241–273, 2004. Reference implementation (C source code): http://jgaa.info/accepted/2004/BoyerMyrvold2004.8.3/planarity.zip

[4] A. Windsor. Planar graph functions for the boost graph library. C++ source code, boost file vault: http://boost-consulting.com/vault/index.php?directory=Algorithms/graph, 2007.

[5] C. Gutwenger and P. Mutzel. Graph embedding with minimum depth and maximum external face. In G. Liotta, editor, *Graph Drawing 2003*, volume 2912 of *LNCS*, pages 259–272. Springer Verlag, 2004.

[6] P. W. Kasteleyn. The statistics of dimers on a lattice: I. the number of dimer arrangements on a quadratic lattice. *Physica*, 27(12):1209–1225, 1961.

[7] M. E. Fisher. Statistical mechanics of dimers on a plane lattice. *Phys Rev*, 124(6):1664–1672, 1961.

[8] A. Globerson and T. Jaakkola. Approximate inference using planar graph decomposition. In B. Schölkopf, J. Platt, and T. Hofmann (eds), *Advances in Neural Information Processing Systems 19*, 2007. MIT Press.

[9] J. Edmonds. Maximum matching and a polyhedron with 0,1-vertices. *Journal of Research of the National Bureau of Standards*, 69B:125–130, 1965.

[10] J. Edmonds. Paths, trees, and flowers. *Canadian Journal of Mathematics*, 17:449–467, 1965.

[11] W. Cook and A. Rohe. Computing minimum-weight perfect matchings. *INFORMS Journal on Computing*, 11(2):138–148, 1999. C source code: http://www.isye.gatech.edu/~wcook/blossom4

[12] N. N. Schraudolph and D. Kamenetsky. Efficient exact inference in planar Ising models. Technical Report 0810.4401, arXiv, 2008. http://aps.arxiv.org/abs/0810.4401

[13] S. V. N. Vishwanathan, N. N. Schraudolph, M. Schmidt, and K. Murphy. Accelerated training conditional random fields with stochastic gradient methods. In *Proc. Intl. Conf. Machine Learning*, pages 969–976, New York, NY, USA, 2006. ACM Press.

[14] B. Taskar, C. Guestrin, and D. Koller. Max-margin Markov networks. In S. Thrun, L. Saul, and B. Schölkopf (eds), *Advances in Neural Information Processing Systems 16*, pages 25–32, 2004. MIT Press.

[15] H. Schramm and J. Zowe. A version of the bundle idea for minimizing a nonsmooth function: Conceptual idea, convergence analysis, numerical results. *SIAM J. Optimization*, 2:121–152, 1992.

[16] C. Rother, V. Kolmogorov, A. Blake, and M. Brown. GrabCut ground truth database, 2007. http://research.microsoft.com/vision/cambridge/i3l/segmentation/GrabCut.htm

[17] S. V. N. Vishwanathan, K. Borgwardt, and N. N. Schraudolph. Fast computation of graph kernels. In B. Schölkopf, J. Platt, and T. Hofmann (eds), *Advances in Neural Information Processing Systems 19*, 2007.

[18] V. Kolmogorov and C. Rother. Minimizing nonsubmodular functions with graph cuts – a review. *IEEE Trans. Pattern Analysis and Machine Intelligence*, 29(7):1274–1279, 2007.
